# L-TTA: Lightweight Test-Time Adaptation Using a Versatile Stem Layer

**Jin Shin, Hyun Kim***
Department of Electrical and Information Engineering and RCEIT
Seoul National University of Science and Technology
Seoul, Korea
{shinjin0103, hyunkim}@seoultech.ac.kr

## Abstract

Test-time adaptation (TTA) is the most realistic methodology for adapting deep learning models to the real world using only unlabeled data from the target domain. Numerous TTA studies in deep learning have aimed at minimizing entropy. However, this necessitates forward/backward processes across the entire model and is limited by the incapability to fully leverage data based solely on entropy. This study presents a groundbreaking TTA solution that involves a departure from the conventional focus on minimizing entropy. Our innovative approach uniquely remodels the stem layer (*i.e.*, the first layer) to emphasize minimizing a new learning criterion, namely, uncertainty. This method requires minimal involvement of the model's backbone, with only the stem layer participating in the TTA process. This approach significantly reduces the memory required for training and enables rapid adaptation to the target domain with minimal parameter updates. Moreover, to maximize data leveraging, the stem layer applies a discrete wavelet transform to the input features. It extracts multi-frequency domains and focuses on minimizing their individual uncertainties. The proposed method integrated into ResNet-26 and ResNet-50 models demonstrates its robustness by achieving outstanding TTA performance while using the least amount of memory compared to existing studies on CIFAR-10-C, ImageNet-C, and Cityscapes-C benchmark datasets. The code is available at https://github.com/janus103/L_TTA.

## 1 Introduction

From the inception of deep learning research, the problem of overfitting, where a model's performance becomes biased towards its training dataset, has been a persistent issue. Various solutions such as normalization and regularization have been proposed to address these issues [26, 56, 30]. Additionally, the need for trained models to adapt to the diversity of data distributions in the real world has also emerged recently. This phenomenon, known as domain shift, has resulted in the proposal of various scenarios and practical studies [63]. Scenarios for addressing model domain shifts are generally introduced with the key themes of domain adaptation (DA) or domain generalization (DG). These are differentiated by their capability to access the data in the target domain. To clarify, DA can access all domains but can only use unlabeled data in the target domain [16, 64, 58, 2], whereas DG can only train in the source domain [66, 34, 49, 35]. Recent source-free DA has indicated that accessing source domain data is generally infeasible. It has proposed a scenario where the target domain adapts using a teacher model that generates pseudo labels [48, 31, 12]. However, this approach involves the significant cost of maintaining the large memory required for the teacher model.

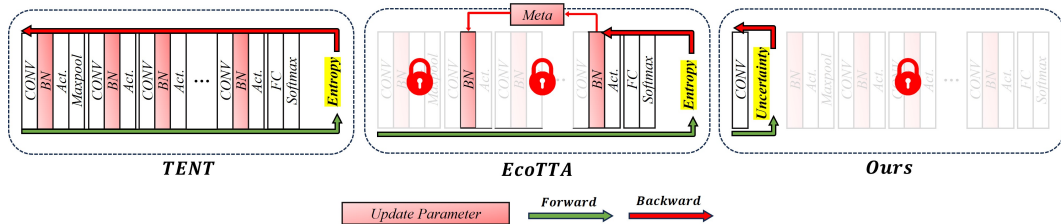

Figure 1: Diagram comparing the forward/backward flow and update process with TENT [59] and EcoTTA [55], illustrated as representative algorithms for entropy minimization and memory-efficient methods, respectively. The red lock icon indicates the absence of TTA execution.

Within our current understanding, the most realistic research topic for deploying deep learning models in the domain shift field at present is test time adaptation (TTA) [39]. This approach relies solely on pre-trained model parameters and data from the target domain without generating pseudo labels. Typically, TTA research focuses on two critical observations: (1) the variation in batch normalization (BN) statistics when domain shift occurs [52, 42] and (2) the possibility of adaptation when the models are trained to minimize the entropy of predictions [59]. Notably, recent studies based on entropy minimization improved the performance by simultaneously passing various augmentations through the input at test-time and either minimizing the average entropy [65] or filtering out noisy data points with high entropy that hinder training [45]. Although these studies enhanced the adaptation performance, they were limited by the substantial training costs and memory usage (which reduced their practicality). For these reasons, TTA performance improvement must be achieved, but in practical terms, there is a need to simultaneously eliminate unnecessary processes and conserve memory resources as much as possible due to gradient, which causes the largest power consumption in training.

We summarize the fundamental problems of such studies into three main aspects: (1) Forwarding the entire model process to obtain the entropy involves a cost. (2) To enable the backward process, a substantial cost would be incurred to reach the initial normalization layer. (3) The absence of a method to ensure data independence may result in biased learning in the incorrect direction if noisy data are not identified, and even if such data are identified, this process can increase the training time.

Our research begins with the hypothesis that fine-tuning the first convolutional (CONV) layer, known as the stem layer, can significantly impact the TTA results. This is based on the understanding that domain shifts in input images affect model outcomes. To overcome the drawbacks of the aforementioned entropy minimization, we introduce a new paradigm for fast and high-performance TTA. This is achieved using a reconstructed stem layer that incorporates a Gaussian channel attention layer (GCAL) constructed on the squeeze and excitation (SE) block [25] and domain embedding layer (DEL) based on a two-dimensional discrete wavelet transform (2D DWT) [29]. Our main contributions can be summarized as follows:

- To implement lightweight TTA, we minimize the uncertainty of channel-wise attention in intermediate features extractable from GCAL rather than an entropy minimization strategy. As depicted in Fig. 1, this involves omitting both forward and backward processes for all the parameters except those in the reconstructed stem layer. Using this stem layer as a pivot point without storing additional gradients provides the fastest training and significantly reduces training costs.

- We propose the integration of DEL as a component within the stem layer to maximize the training effect of a single data point. 2D DWT in the stem layer helps collect domain-invariant edge information efficiently and improves model generalization by providing multi-views with redundant content information before CONV operations. Furthermore, forwarding features extracted from multiple frequency domains to GCAL enables the calculation of uncertainty on an individual basis, thereby maximizing data utilization.

- Additionally, DEL includes inverse DWT (IDWT), thereby enabling a non-invasive design. This yields a shape identical to the intermediate feature obtained from the existing stem layer's CONV operation. This, in turn, enables a convenient integration into any model utilizing a convolutional neural network (CNN)-based backbone for image classification and a wide range of applications (*i.e.*, high scalability).

Experimentally, our method surpasses the state-of-the-art (SOTA) with a 5.4% higher prediction accuracy on CIFAR-10-C, including 15 types of corruptions. On the large dataset, ImageNet-C [22], it displays an accuracy lower than that of the SOTA model by 2.7%; however, it achieves a significant performance with only 1.7% of memory usage compared to the SOTA model. Furthermore, compared with EcoTTA [55], which is the SOTA in terms of memory efficiency, our approach uses 5.9% of memory and displays a marginal decrease in accuracy, by only 0.3%. These results highlight our method's effectiveness in balancing performance and memory efficiency. In terms of scalability, compared with EcoTTA, our method achieved a 4.6% improvement in mean intersection over union (mIoU) with 0.04% of memory usage in a semantic segmentation task under four weather conditions (*e.g.*, brightness, foggy, rainy, and snow).

## 2   Backgrounds

### 2.1   Domain Shift Scenarios

Deep learning models learn from high-dimensional data such as images and natural language. These models are influenced significantly by the data distribution [18]. Data derived from the source domain, denoted as $D_s = \{(x_i^s, y_i^s)\}_{i \in \{0...N-1\}}$, and target domain, denoted as $D_t = \{(x_i^t, y_i^t)\}_{i \in \{0...N-1\}}$, generally exhibit significantly different distributions. Therefore, it could be challenging for a model trained and fitted on data from $D_s$ to infer data from $D_t$. Scenarios designed to address this domain shift are classified into DG and DA depending on whether there is direct access to the $D_t$ distribution during training. Both strategies primarily aim to align the feature space with domain-invariant content information and style information sensitive to variations. In DG, where access to $D_t$ is unavailable, auxiliary networks are frequently employed to learn the adjustment of means and standard deviations to fit the domain [15] or use generative models to create $D_t$ adversarial to the distribution of $D_s$ and train simultaneously [49, 35]. Recently, multiple augmentations were employed in conjunction with an equal number of meta-networks to comprehend meta-knowledge regarding the domain directly. This secured superior generalization performance [6]. However, this competitive trend increases the load regarding training costs and inference, thereby necessitating improvements in practicality and efficiency. In DA, where direct access to $D_t$ is feasible, model generalization is more convenient. However, a significant characteristic is the unavailability of labels. Early DA research involved training with labeled data from $D_s$ [16, 58, 64]. However, in practical applications, storing and utilizing the $D_s$ dataset in memory is considered unrealistic. To overcome this issue, source-free DA employs pseudo-labeling techniques by dually leveraging pre-trained models [48, 31]. Nonetheless, these approaches still have the burden of maintaining the dual models in real-world application deployment as well as the problem of confirmation bias.

### 2.2   Test Time Adaptation

Like DG and DA research, TTA is a scenario developed to solve domain shifts. The focus is on more efficient learning during test-time [39]. TTA essentially assumes training using only data $x_i^t$ without the corresponding label $y_i^t$. Within these constraints, TTA adapts to the distribution of $D_t$ by partially fine-tuning the layers of a pre-trained model.

BN STAT [42] indicates the widespread covariate shift across all the domains that need to be addressed and adjusts the fixed parameters of BN layers, specifically, the mean and variance, at test-time. TENT [59] is the baseline for many TTA approaches. It proposes a strategy to minimize the prediction entropy and significantly improve the performance by updating statistics and affine transformation parameters for each batch. MEMO [65] applies multiple augmentations to the input at test-time and minimizes the average (marginal entropy) of entropies obtained by passing the model through, thereby demonstrating better adaptation effects. EATA [45] extends TENT as the baseline. It expounds that data points with high entropy do not contribute to adaptation and proposed criteria for establishing a threshold. SAR [46] advances further by addressing realistic mixed distribution variations, small batch sizes, and imbalanced label distribution variations. It aims for a stable TTA by allocating the same class to all the samples and proposes a strategy to minimize the sharpness of the loss surface. REALM [53] achieves the highest performance among entropy minimization-based studies. Unlike other methodologies, REALM introduces a framework based on self-supervised learning that enables the inclusion of noise samples in training without skipping. However, based on existing research, achieving TTA through entropy minimization incurs the basic training cost for performing

forward/backward operations globally in the model. Moreover, it has limitations in assessing whether the data is suitable for TTA through entropy.

For these reasons, there is a growing interest in whether the proposed methods can be applied efficiently in the real world. In particular, EcoTTA [55] introduces a pioneering method to minimize memory consumption during the TTA process. This is achieved by partially placing auxiliary networks for TTA outside the main network. The approach ensures that only the BN layers included in the main and auxiliary networks pass through during backpropagation, significantly shortening the path. This simultaneously reduces the memory required for gradients and the time consumed. MECTA [24] follows a training flow similar to EcoTTA, where all layers of the model are forwarded and a specialized normalization layer is introduced to minimize memory usage, particularly within the cache. However, it relies on methodologies like TENT and EATA to perform TTA. SFT [33] based on MEMO divides the model into subsets and introduces a method of learning only certain layers based on the distribution of $D_t$. This provides insight into the fact that learning the initial layers is more effective in supporting adaptation learning than the subsequent ones. Notwithstanding these efforts, it is necessary to forward the entire model for TTA, owing to the training based on entropy. Moreover, during backpropagation, the problem of reaching batch normalization at the beginning of the backbone remains while the gradients of many layers are still being calculated.

Therefore, approaches that can adapt at test time without relying on entropy minimization have also been introduced. DDA [17] has proposed a method that directly projects inputs obtained from $D_t$ to $D_s$ using a generative model without performing expensive retraining. However, this can yield more expensive results in maintaining additional systems, similar to augmentation.

## 2.3 Discrete Wavelet Transform

The wavelet transform (WT) provides a flexible time–frequency resolution by analyzing signals at multi scales, making it a more effective tool than the Fourier transform in signal processing [4]. Moreover, owing to the short and localized characteristics of wavelet functions, WT is suitable for processing non-stationary signals [54]. Based on these advantages, 2D DWT [28] is commonly used to extract detail and edge information in the spatial domain. Additionally, it allows for multi-level decomposition until both the width and height of the input image are powers of two. This enables aggressive summarization and extraction of essential information. Through the inverse transform, the original image can be reconstructed highly accurately with minimal information loss. The 2D DWT process involves sequentially applying a one-dimensional DWT (1D DWT) [44] to the rows and columns of the input data. For 1D DWT, we operate using the simplest wavelet family, *i.e.*, the Haar wavelet [36]. It should be noted that Haar can be performed conveniently through CONV operations [47]. In the first stage, the horizontal direction decomposition is performed to split a single original feature map into two components: the low-frequency component (LFC), which generally contains visually intuitive information and represents the basic image structure, and the high-frequency component (HFC), which includes finer details of the image and a small amount of edge information. In the second stage, a vertical decomposition is performed as the two filters are transposed, as illustrated in Fig. 2. The detailed 2D DWT process is presented in Appendix B.

## 3 Proposed Method

### 3.1 Overview

**Motivation:** To improve the performance of TTA, numerous studies have focused on efficiently minimizing the entropy of model predictions obtained from unlabeled data $x_i^t$. Although this has enhanced the accuracy, efforts to reduce the memory consumption, which significantly impacts the power consumption, and perform TTA faster have been relatively few. Nevertheless, these research areas are essential for applying TTA in real-world scenarios. In practice, using augmentation in image pre-processing during training, dynamically filtering data advantageous for training, or replicating pre-trained weights generates a significant load. Furthermore, we identify a key drawback in most existing TTA studies that adopt entropy. Even without parameter updates, calculating entropy necessitates forward/backward passes through the entire model layers. This imposes a fundamental training cost. This limitation motivates us to resolve the existing issues in TTA.

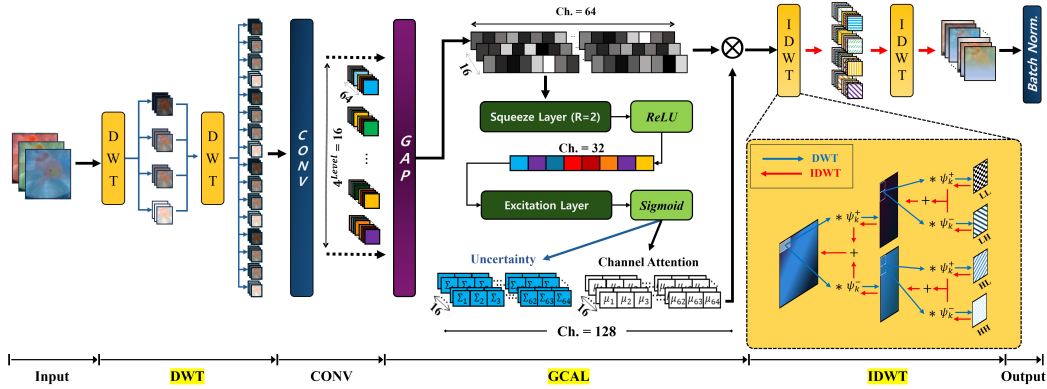

Figure 2: Overview of our method including the reconstructed stem layer. $\otimes$ and $*$ denote element-wise multiplication and CONV operation, respectively, while $\psi_k^+$ and $\psi_k^-$ respectively denote the low and high filters for DWT, described in Appendix B.

**Goal:** To eliminate existing drawbacks and minimize training costs, we aim to provide an alternative to entropy minimization as a TTA solution. Specifically, our design goals are summarized as follows: 1) *Practicality in training*: Minimizing the resources required for training, such as memory and data, to attain acceptable a reasonable prediction accuracy in $D_t$. 2) *Scalability*: Designing to be non-invasively and conveniently applicable in CNN-based tasks without modifying other layers. 3) *Data leveraging*: Maximizing usability from independent data to achieve TTA within constraints, even with small batch sizes or a single batch.

**Approach:** The sequential influence of CNN layers results in significant variations in prediction quality across different input image domains. By fine-tuning the first representation of the input image, fast adaptation to $D_t$ is possible. Therefore, instead of expensive entropy, we extract and minimize the channel-wise uncertainty from the reconstructed stem layer to adapt to $D_t$.

**Workflow:** As shown in Fig. 2, the reconstructed stem layer comprises the original CONV layer and two key architectures in this study, GCAL and DEL. The reconstructed stem layer follows the sequence of DWT, CONV, GCAL, and IDWT. The processes of DWT and IDWT are encompassed in DEL. As depicted in Fig. 2, GCAL is the only key architecture that enables adaptation and outputs channel-wise attention and its associated uncertainty in an end-to-end manner. Similarly, as with an SE block, the extracted attention is scalar and applied to each feature on a per-channel basis. Meanwhile, the uncertainty is minimized through negative log-likelihood (NLL) loss. In DEL, DWT decomposes the input feature map into multiple frequency domains while maintaining spatiality. This step is performed before CONV. It allows for capturing more diverse features from independent data, thereby better grasping the differences between $D_s$ and $D_t$. By performing IDWT at the end of the stem layer, the shape of the features is restored to its original state before the stem layer is modified. This enables the non-invasive application of the stem layer in pre-trained models and synthesizes important redundant information from multiple views to enhance the generality. While injecting the stem layer into a pre-trained model for warm-up, we jointly train it to reduce task-specific prediction errors using suitable loss terms (*i.e.*, cross-entropy) in conjunction with the uncertainty extracted from GCAL minimized through NLL loss. It should be noted that since uncertainty is minimized only during training, in the TTA setting, the procedures after IDWT are not required.

## 3.2 GCAL: Gaussian Channel Attention Layer

GCAL is a key architecture that enables TTA using only the stem layer. It remodels the SE block to predict uncertainty. Please note that the SE block [25] is not incorporated into other layers and is trained end-to-end. It dynamically extracts channel-wise attention $\gamma_{scale}$ defined below from intermediate features and performs recalibration. This enhances the representational capacity of the CONV and, ultimately, the prediction accuracy.

$$\gamma_{scale} = F_{se}(\tilde{X}, W) = \sigma(W_2 \delta(W_1, \tilde{X})), \tag{1}$$

where $\tilde{X}$ represents the intermediate feature obtained after performing global average pooling, which in ResNet [20] comprises 64 channels. The weight $W_1$ reduces the channel size of $\tilde{X}$ based on a predetermined hyperparameter, whereas $W_2$ conversely expands it back to its original channel size. $\sigma$ and $\delta$ denote the sigmoid and ReLU functions [43], respectively.

In Gaussian Yolov3 [8, 9], rather than predicting variables such as the bounding box coordinates, the model is designed to extract the mean and variance of a Gaussian distribution to quantify the uncertainty. Similarly, in this study, the output of the SE block, $\gamma_{scale}$, is represented by Gaussian parameters, namely, mean $\gamma_\mu$ and variance $\gamma_\Sigma$. The probability density function for quantifying the uncertainty of $\gamma_{scale}$ is defined as follows:

$$p(\mu; \gamma_\mu, \gamma_\Sigma) = \sqrt{\frac{1}{2\pi\gamma_\Sigma}} e^{-\frac{(\gamma_\mu - \mu)^2}{2\gamma_\Sigma}}. \tag{2}$$

For implementation, without significantly altering the $F_{se}$ structure, we obtain $\gamma_\mu$ and $\gamma_\Sigma$ by increasing the channel size by 100%. This is shown in Fig. 2. These two Gaussian parameters are defined as follows:

$$\gamma_\mu = \sigma(W_2^\mu \delta(W_1, \tilde{X})), \quad \gamma_\Sigma = \sigma(W_2^\Sigma \delta(W_1, \tilde{X})). \tag{3}$$

To perform TTA, we minimize the Gaussian parameter $\gamma_\Sigma$ to reduce the uncertainty of $\gamma_{scale}$, which is multiplied channel-wise. It should be noted that $\gamma_{scale} = \gamma_\mu$. However, because $\gamma_\mu$ is determined dynamically based on the input, it is challenging to determine the ground-truth (GT) $\mu$ in the TTA setting. Therefore, we aim to modify the SE block to function primarily as an extractor of uncertainty for TTA. In all the training settings, we set the GT of $\gamma_\mu$ to the maximum value of the sigmoid function ($= 1$) to enable training. Reflecting this approach, we redefine the NLL loss to minimize $\gamma_\Sigma$ (an uncertainty) in conjunction with Eq. 2:

$$L_{uncertainty} = \frac{1}{C} \sum_{i=0}^{C-1} -log(p_i(\mu_{gt}; \gamma_\mu, \gamma_\Sigma)), \tag{4}$$

where $C$ represents the total number of channels in intermediate features. Eq. 4 means minimizing uncertainty for all channels, which applies equally to pretraining and TTA processes.

### 3.3 DEL: Domain Embedding Layer

In recent TTA scenarios, the entropy minimization model is traditionally learned by filtering out data that is evaluated to have low entropy. However, contrary to this trend, [61] experimentally shows that HFC derived from input images helps generalize the model. This means that high entropy actually contributes significantly to improving prediction accuracy. This entropy ambiguity must be avoided to improve the individual usability of single input data $x^t$. However, to solve this problem, it is clearly challenging to determine the entropy threshold and control strength without additional training costs at test time.

To alleviate entropy ambiguity, we propose DEL, which encapsulates GCAL and CONV layers with DWT and IDWT layers, as shown in Fig. 2. This decomposes the independent $x^t$ into various frequency domains and allows for end-to-end learning of uncertainty for each channel. Accordingly, we redefine the loss term for uncertainty with Eq. 4 as follows:

$$L_{uncertainty} = \frac{1}{C}\frac{1}{N} \sum_{i=0}^{C-1} \sum_{j=0}^{N-1} -log(p_{i,j}(\mu_{gt}; \gamma_\mu, \gamma_\Sigma)), \tag{5}$$

where N represents the number of frequencies decomposed into DWT. As shown in Fig. 3(a), the DWT layer maintains spatiality without resynthesis even after frequency decomposition, unlike the well-known transformation method [4, 51], even after decomposition. This makes learning spatial structures, such as edges or patterns, possible without additional computation. The IDWT layer, located last, makes the input of the subsequent layer have the same shape as when there is no DEL and maintains the spatiality of the feature. As a result, DEL enables non-invasive design.

Furthermore, we propose omnidirectional decomposition (ODD), which additively decomposes LFC and HFC simultaneously. As shown in Fig. 3(a), when DWT is performed once, it can be observed that edge information overlapping with LFC still remains in HFC. We decompose edge and noisy

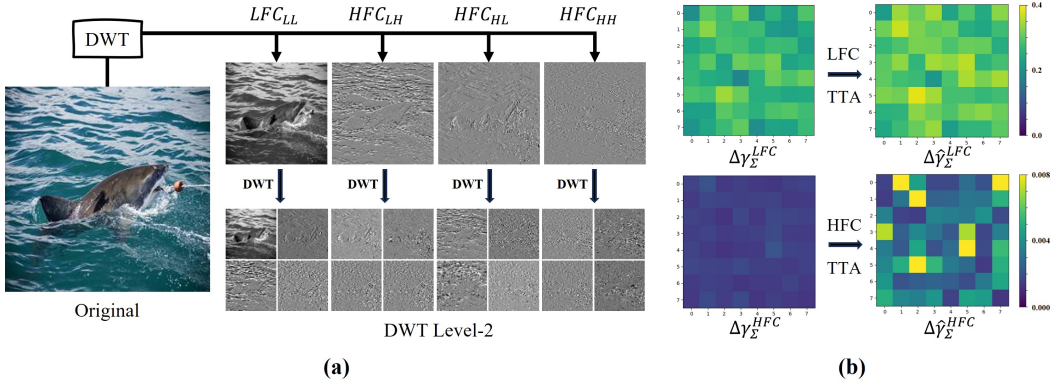

(a)                                                    (b)

Figure 3: (a) visualizes the proposed ODD process in detail on a given input. (b) shows the variance of uncertainty according to the domain shift of LFC (=$LFC_{LL}$) and HFC (=$HFC_{LH}$).

information as much as possible at a level that maintains spatiality, making it possible to calculate the uncertainty of a single input $x^t$ more sensitively. Additionally, as a side effect, generalization performance is further enhanced by the model concretely identifying noisy data contained in HFC.

Fig. 3(b) visualizes the changes in the uncertainty map for each of the 64 channels for LFC and HFC as we perform our proposed TTA. Denoted $\Delta\gamma_{\Sigma}$ and $\Delta\hat{\gamma}_{\Sigma}$ are defined as follows:

$$\Delta\gamma_{\Sigma} = |log(F_{se}(\tilde{X}_s; W) - F_{se}(\tilde{X}_t; W))|, \ \Delta\hat{\gamma}_{\Sigma} = |log(F_{se}(\tilde{X}_s; W) - F_{se}(\tilde{X}_t; \hat{W}))|. \quad (6)$$

In $\Delta\gamma_{\Sigma}^{LFC}$, there is a clear difference in the uncertainty for each channel obtained with $\tilde{X}_s$ and $\tilde{X}_t$ even before performing TTA. On the other hand, $\Delta\gamma_{\Sigma}^{HFC}$ shows that $F_{SE}(\cdot)$ does not have the capability to properly obtain uncertainty about $\tilde{X}_t$. Therefore, we prove through $\Delta\hat{\gamma}_{\Sigma}^{HFC}$ that uncertainty can be correctly extracted by performing TTA.

### 3.4 Interval Training for Continual Setting

For performance assessment of TTA, achieving high prediction performance with low training cost is important, but it is also important to design it to enable continuous training. To overcome catastrophic forgetting, existing studies [55, 62] make this possible by maintaining a teacher model of the same shape as the target model or applying regularization techniques for each layer. Instead of these complicated strategies, we reload the weights of the pre-trained stem layer and update them according to $D_t$ at every short interval. This is an efficient scenario when initialized weights consume less memory and simultaneously perform fast TTA, and the proposed reconstructed stem layer realizes this. The short interval setting allows for natural adaptation even when the domain shifts frequently and preserves the generalization performance of the pre-trained model.

## 4 Experimental Results

In this section, we quantitatively and qualitatively demonstrate three contributions related to TTA obtained by applying stem layers reconstructed by DEL and GCAL to CNN-based models. We evaluate the practicality in training (*i.e.*, comparison of accuracy and memory efficiency with SOTA TTA methods in Tab. 1 and Fig. 4), scalability (*i.e.*, applicability on image classification and semantic segmentation in Tab. 2), and data leveraging (*i.e.*, assessment of the TTA performance at small batch sizes in Fig. 7). In Appendices C,D,E and F, we further rigorously evaluate the proposed method by extending the model, using challenging datasets, and exploring generalization scenarios (*i.e.*, source-free DA), along with an ablation study for hyperparameters.

### 4.1 Implementation Details

In all the experimental settings, the model backbones use ResNet-26 and ResNet-50 [20]. These are pre-trained with ImageNet [14]. To evaluate the robustness of the model in the existing TTA setting,

Table 1: Comparisons of prediction error (%) with prior TTA methods on CIFAR-10-C with ResNet-26 and ImageNet-C with ResNet-50.

| Method | Noise | | | Blur | | | | Weather | | | | Digital | | | | Avg. |
|---|---|---|---|---|---|---|---|---|---|---|---|---|---|---|---|---|
| | Gauss. | Shot | Impul. | Defoc. | Glass | Motion | Zoom | Snow | Frost | Fog | Brit. | Contr. | Elastic | Pixel | JPEG | |
| CIFAR-10-C | | | | | | | | | | | | | | | | |
| Source | 86.9 | 82.6 | 81.8 | 11.4 | 50.2 | 18.9 | 9.1 | 16.4 | 26.4 | 18.4 | 7.1 | 24.5 | 22.8 | 64.0 | 28.3 | 36.6 |
| **Ours w/o GCAL** | 62.4 | 55.4 | 48.5 | 12.1 | 51.7 | 15.5 | 8.5 | 16.6 | 26.4 | 21.7 | 6.4 | 31.0 | 21.3 | 61.7 | 22.5 | 30.8 |
| TENT[59] | 39.4 | 38.8 | 47.9 | 19.9 | 45.0 | 23.2 | 20.6 | 28.1 | 32.1 | 24.5 | 16.1 | 26.7 | 32.4 | 30.6 | 35.5 | 30.7 |
| MEMO[65] | 43.5 | 39.9 | 43.3 | 26.4 | 44.4 | 25.1 | 25.0 | 20.9 | 28.3 | 22.8 | 11.9 | 28.3 | 21.1 | 42.8 | **21.7** | 29.7 |
| SFT[33] | 31.9 | 26.7 | **28.9** | 17.7 | 44.2 | 18.4 | 20.2 | 20.8 | 23.4 | 20.7 | 13.9 | 25.4 | 24.5 | 21.9 | 25.1 | 24.2 |
| EATA[45] | 33.9 | 32.8 | 41.4 | 19.4 | 42.4 | 20.5 | 20.1 | 22.4 | 27.1 | 22.7 | 13.8 | 24.0 | 24.0 | 29.3 | 26.5 | 26.7 |
| SAR[46] | 46.4 | 40.9 | 50.1 | 20.2 | 47.0 | 21.7 | 20.8 | 22.9 | 29.5 | 23.9 | 13.8 | 25.5 | 24.3 | 39.5 | 27.4 | 30.3 |
| REALM[53] | **27.8** | **25.4** | 35.5 | 15.5 | 37.7 | 17.4 | 16.9 | 20.4 | 22.3 | 19.0 | 12.9 | 18.0 | 23.1 | 22.0 | 24.2 | 22.5 |
| **Ours w/o DEL** | 33.1 | 29.1 | 31.3 | 10.0 | 36.9 | **11.4** | 7.3 | **12.8** | **13.1** | **11.6** | **5.7** | **6.9** | **18.4** | 15.7 | 24.5 | 17.8 |
| **Ours** | 31.0 | 26.8 | 30.5 | **9.2** | **34.9** | **11.4** | **7.1** | 13.1 | 13.8 | 12.2 | **5.7** | 7.5 | **18.4** | **12.6** | 22.6 | **17.1** |
| ImageNet-C | | | | | | | | | | | | | | | | |
| Source | 97.8 | 97.1 | 98.2 | 82.1 | 90.2 | 85.2 | 77.5 | 83.1 | 76.7 | 75.6 | 41.1 | 94.6 | 83.1 | 79.4 | 68.4 | 82.0 |
| **Ours w/o GCAL** | 82.7 | 82.6 | 86.2 | 79.1 | 89.1 | 84.2 | 75.7 | 74.8 | 67.3 | 73.1 | 37.5 | 84.6 | 74.1 | 43.2 | 55.2 | 72.6 |
| TENT[59] | 97.5 | 97.1 | 97.5 | 86.5 | 96.4 | 81.4 | 82.4 | 84.7 | 77.0 | 98.6 | 29.6 | 57.8 | 93.8 | 50.8 | 46.2 | 78.5 |
| MEMO[65] | 81.5 | 79.5 | 81.6 | 82.9 | 87.4 | 78.2 | 73.1 | 59.6 | **53.0** | 65.6 | 30.5 | 63.5 | 80.8 | 67.9 | 46.7 | 68.8 |
| DDA[17] | **57.6** | **56.7** | **57.7** | 83.4 | **80.4** | 78.1 | 74.0 | 64.3 | 59.9 | 86.3 | 38.8 | 74.8 | 62.5 | 53.4 | 45.9 | 64.9 |
| EATA[45] | 75.2 | 71.7 | 74.3 | 81.9 | 82.7 | 71.5 | 70.7 | 55.5 | 55.7 | 58.4 | 29.1 | **55.4** | 73.0 | 53.2 | 44.3 | 63.5 |
| SAR[46] | 76.6 | 73.4 | 76.1 | 81.6 | 84.6 | 71.4 | 69.6 | 55.1 | 55.3 | 74.3 | **27.7** | 55.5 | 85.2 | 53.0 | **43.9** | 65.5 |
| REALM[53] | 73.1 | 70.1 | 72.0 | 81.6 | 81.8 | **70.4** | 68.9 | **54.4** | 56.4 | 54.5 | 28.8 | 55.6 | 71.1 | 50.3 | 44.5 | **62.2** |
| **Ours w/o DEL** | 75.8 | 75.0 | 76.7 | 85.1 | 88.3 | 76.0 | 68.1 | 62.0 | 58.7 | **53.0** | 30.3 | 82.7 | 63.4 | 62.5 | 58.0 | 67.7 |
| **Ours** | 79.4 | 78.8 | 82.2 | **75.8** | 81.1 | 72.4 | **63.5** | 68.8 | 58.5 | 53.5 | 33.4 | 73.1 | **57.0** | **41.4** | 54.5 | 64.9 |

we use the benchmark dataset with the well-established 15 types of corruptions (*e.g.*, noise, blur, weather, and digital) [22]. Each corruption is applied to the validation set of the original dataset and has identical content information. These are called CIFAR-10-C, CIFAR-100-C, and ImageNet-C, respectively. The severity of the corruption is divided into five levels. We conventionally apply the most severe level ($= 5$) for an unbiased comparison with the other methods. Additionally, to measure the memory reliably, we refer to the officially provided code of TinyTL [5]. It is identical to that measured by EcoTTA [55]. This considers the memory for storing model parameters and gradients. The gradient size increases exponentially according to the batch size. A detailed evaluation setting of image classification and semantic segmentation models is presented in Appendix A.

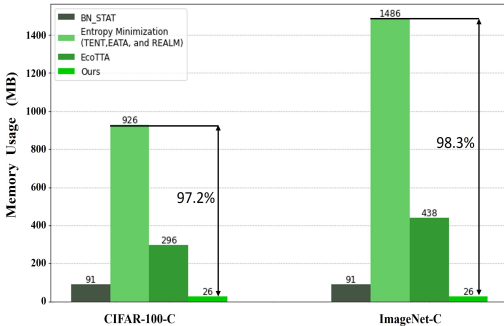

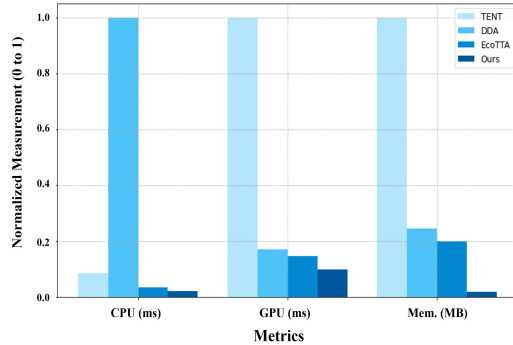

Figure 4: Comparison of memory usage in a single iteration on CIFAR-100-C and ImageNet-C datasets.

Figure 5: Training time comparison of various TTA methods on ResNet-50.

## 4.2 Main Results

Tab. 1 reports prediction errors with regard to accuracy for the image classification task. As mentioned in Sec. 3, DEL included in the stem layer can display generalization performance without additional

Table 2: Comparison of mIoU (%) for semantic segmentation in Cityscapes-C using DeepLabV3Plus with prior TTA methods.

| Method | Bright. | Foggy | Frost | Snow | Avg. | Mem. (MB) |
|---|---|---|---|---|---|---|
| Source | 60.4 | 54.3 | 30.0 | 4.1 | 37.2 | - |
| Ours w/o GCAL | 69.9 | 62.9 | 47.2 | 41.6 | 55.4 | - |
| BN STAT[42] | 69.1 | 61.0 | 44.8 | 39.1 | 53.5 | 280 |
| TENT[59] | 70.1 | 62.1 | 46.1 | 40.2 | 54.6 | 2721 |
| EcoTTA[55] | 70.2 | 62.4 | 46.3 | 41.9 | 55.2 | 918 |
| Ours w/o DEL | **72.0** | **65.7** | 49.6 | 45.3 | 58.1 | **0.4** |
| Ours | 70.6 | 63.5 | **53.1** | **52.0** | **59.8** | **0.4** |

assumptions regarding learning, and this exhibits an average improvement of 5.8% and 9.4% in CIFAR-10-C and ImageNet-C, respectively, compared with the baseline (*i.e.*, source - Ours w/o GCAL). It should be noted that we apply the proposed interval training to meet the continual TTA condition. In these experiments, the weights of the reconstructed stem layer are re-initialized to the pre-trained weights every 20 iterations for CIFAR-10-C and every 10 iterations for ImageNet-C. As shown in Tab. 1, for CIFAR-10-C, it shows a 5.4% higher improvement on average than SOTA. Meanwhile, although the results of ImageNet-C in Tab. 1 do not exceed the SOTA performance, it achieves accuracy comparable to SOTA with significantly less memory usage as can be seen in Fig. 4. In addition, it is noteworthy that the proposed method achieves 3.9% better performance than MEMO [65], which performs entropy minimization using various augmentations in ImageNet-C.

Ultimately, the goal of injecting the stem layer is to demonstrate an efficient TTA scenario that considers the trade-off between efficient prediction and memory consumption, and our method displayed an overwhelming performance advantage in terms of memory-saving compared with prior works. In Fig. 4, we illustrate the memory usage for ResNet-50, with a focus on BN STAT, entropy minimization-based research, and memory-saving EcoTTA [55]. Herein, TENT [59], EATA [59], and REALM [53] displayed equal memory consumption with the result of entropy minimization-based research. Compared to these three approaches, our proposed approach consumes only 28.6%, 2.8%, and 8.8% memory for CIFAR-100-C and 28.6%, 1.7%, and 5.9% memory for ImageNet-C, respectively, for performing TTA.

In Fig. 5, we compare the proposed method with three representative approaches from entropy minimization-based TTA (*i.e.*, TENT), diffusion-driven TTA (*i.e.*, DDA), and memory-efficient TTA (*i.e.*, EcoTTA) methodologies in terms of latency on a general heterogeneous system (*i.e.*, CPU and GPU). Our method demonstrates $4.25\times$, $49.56\times$, and $1.76\times$ faster performance in total latency, respectively, indicating that the proposed L-TTA achieves significantly faster training times than existing TTA methods. It is also important to note that since TENT and EcoTTA perform the TTA process using a single independent model, their latency tends to depend on memory usage. In contrast, DDA shows considerable CPU usage due to the image preprocessing tasks such as noise synthesis, which are not handled by the GPU. These CPU-intensive processes can lead to substantial power consumption unless dedicated hardware is available. Therefore, despite lower memory usage, using such methodologies for field applications should be avoided as much as possible.

The proposed stem layer can be conveniently expanded regardless of the task because only the initial parts of the entire network are changed. To demonstrate this, Tab. 2 provides additional results in the semantic segmentation task. Even without performing TTA (*i.e.*, Ours w/o GCAL), mIoU is improved by an average of 18.2% and 0.8% compared with the source model and TENT, respectively. When TTA is applied (*i.e.*, Ours), it improves mIoU by 4.6% compared to EcoTTA with only 0.04% of memory usage.

## 4.3 Ablation Studies

**Interval adjustment in the continual scenario.** As discussed in Subsection 3.4, we propose interval training to satisfy the continuous TTA scenario. Accordingly, we conduct an ablation study to observe the prediction error variations using the ResNet-26 model, as reported in Tab. 1. In this experiment, we adjust the iteration interval for performing one round of the TTA process and observe the resulting

prediction error. In Fig. 6, *0* and *Full iter.* on the x-axis represent the cases of no TTA and completing 1 epoch, respectively, with the intermediate values showing results for iteration increments of 5 steps. The results demonstrate that applying GCAL and DEL to the stem layer consistently yields better outcomes, with a steady decline in error rate as iterations increase. Furthermore, both methods also outperform REALM, a SOTA approach, after 10 iterations. Notably, the proposed L-TTA (=DEL+GCAL) achieves a 5.7% performance improvement after completing 1 epoch.

**Evaluation on small batch sizes.** By implementing DWT in DEL, we provide a foundation for generalization and maximize our capabilities by minimizing individual uncertainties across various frequency domains with spatial properties. We contend that this approach assists data in conveniently adapting independently to the target domain. In Fig. 7, we present the results of performing TTA on ImageNet-C for small batch sizes (=1, 2, 4, 8), in conjunction with the representative methods of entropy minimization (*i.e.*, TENT [59], EATA [45] and SAR [46]). The conventional methods exhibit a near-complete loss of classification capability, with an average prediction error of 99.89% when the batch size is one. Meanwhile, our proposed method demonstrates a higher performance, with a prediction error of 75.16%. Additionally, it is visually evident that our method is less sensitive to variations in batch size, with a difference of 8.37% between batch sizes of one and eight, whereas TENT, EATA and SAR exhibit instability with differences of 31.45%, 22.67% and 30.86%, respectively. This is interpreted as a result of the significant dependency of the conventional methods on BN layers.

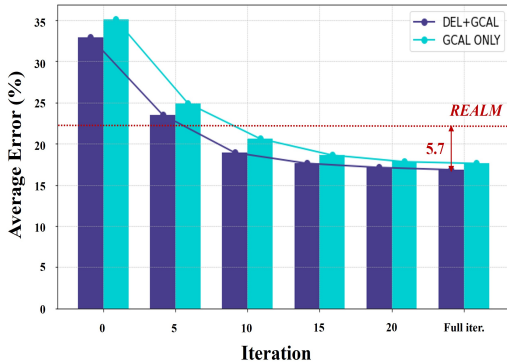
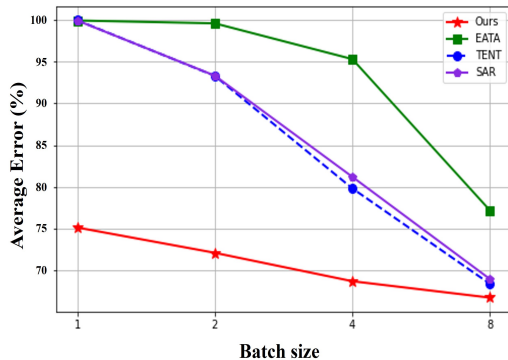

Figure 6: Comparison of prediction error (%) across increasing TTA iterations on CIFAR-10-C with ResNet-26.

Figure 7: Assessment of TTA effectiveness for small batch sizes in comparison with entropy minimization baselines on ImageNet-C.

## 5   Conclusion

We propose a novel method for lightweight TTA that utilizes the stem layer reconstructed with DEL and GCAL to minimize the uncertainty across multiple frequency domains rather than the entropy. This architecture improves the model's generalization performance without assumptions regarding training and can be applied conveniently and non-invasively to models with a CNN as the backbone. Our approach demonstrates a performance equivalent to or superior to those in SOTA studies with well-known robust benchmark datasets, showcasing the fastest and most memory-efficient results suitable for real-world scenarios.

## Acknowledgement

This research was partly supported by the MSIT(Ministry of Science and ICT), Korea, under the ITRC(Information Technology Research Center) support program (IITP-2024-RS-2022-00156295) supervised by the IITP(Institute for Information & Communications Technology Planning & Evaluation) and Basic Science Research Program through the National Research Foundation of Korea(NRF) funded by the Ministry of Education(NRF-2019R1A6A1A03032119).

## Footnotes

*Corresponding author: Hyun Kim, email: hyunkim@seoultech.ac.kr

# References

[1] Shun-ichi Amari. Backpropagation and stochastic gradient descent method. *Neurocomputing*, 5(4-5):185–196, 1993.

[2] Yiyao An, Ke Zhang, Yi Chai, Qie Liu, and Xinghua Huang. Domain adaptation network base on contrastive learning for bearings fault diagnosis under variable working conditions. *Expert Systems with Applications*, 212:118802, 2023.

[3] Malik Boudiaf, Romain Mueller, Ismail Ben Ayed, and Luca Bertinetto. Parameter-free online test-time adaptation. In *Proceedings of the IEEE/CVF Conference on Computer Vision and Pattern Recognition*, pages 8344–8353, 2022.

[4] E Oran Brigham. *The fast Fourier transform and its applications*. Prentice-Hall, Inc., 1988.

[5] Han Cai, Chuang Gan, Ligeng Zhu, and Song Han. Tinytl: Reduce memory, not parameters for efficient on-device learning. *Advances in Neural Information Processing Systems*, 33:11285–11297, 2020.

[6] Jin Chen, Zhi Gao, Xinxiao Wu, and Jiebo Luo. Meta-causal learning for single domain generalization. In *Proceedings of the IEEE/CVF Conference on Computer Vision and Pattern Recognition*, pages 7683–7692, 2023.

[7] Liang-Chieh Chen, Yukun Zhu, George Papandreou, Florian Schroff, and Hartwig Adam. Encoder-decoder with atrous separable convolution for semantic image segmentation. In *Proceedings of the European conference on computer vision (ECCV)*, pages 801–818, 2018.

[8] Jiwoong Choi, Dayoung Chun, Hyun Kim, and Hyuk-Jae Lee. Gaussian yolov3: An accurate and fast object detector using localization uncertainty for autonomous driving. In *Proceedings of the IEEE/CVF International conference on computer vision*, pages 502–511, 2019.

[9] Jiwoong Choi, Dayoung Chun, Hyuk-Jae Lee, and Hyun Kim. Uncertainty-based object detector for autonomous driving embedded platforms. In *2020 2nd IEEE International Conference on Artificial Intelligence Circuits and Systems (AICAS)*, pages 16–20, 2020.

[10] Sungha Choi, Sanghun Jung, Huiwon Yun, Joanne T Kim, Seungryong Kim, and Jaegul Choo. Robustnet: Improving domain generalization in urban-scene segmentation via instance selective whitening. In *Proceedings of the IEEE/CVF Conference on Computer Vision and Pattern Recognition*, pages 11580–11590, 2021.

[11] Sungha Choi, Seunghan Yang, Seokeon Choi, and Sungrack Yun. Improving test-time adaptation via shift-agnostic weight regularization and nearest source prototypes. In *European Conference on Computer Vision*, pages 440–458. Springer, 2022.

[12] Dayoung Chun, Seungil Lee, and Hyun Kim. Usd: Uncertainty-based one-phase learning to enhance pseudo-label reliability for semi-supervised object detection. *IEEE Transactions on Multimedia*, 2024.

[13] Marius Cordts, Mohamed Omran, Sebastian Ramos, Timo Rehfeld, Markus Enzweiler, Rodrigo Benenson, Uwe Franke, Stefan Roth, and Bernt Schiele. The cityscapes dataset for semantic urban scene understanding. In *Proc. of the IEEE Conference on Computer Vision and Pattern Recognition (CVPR)*, 2016.

[14] Jia Deng, Wei Dong, Richard Socher, Li-Jia Li, Kai Li, and Li Fei-Fei. Imagenet: A large-scale hierarchical image database. In *2009 IEEE Conference on Computer Vision and Pattern Recognition*, pages 248–255, 2009.

[15] Xinjie Fan, Qifei Wang, Junjie Ke, Feng Yang, Boqing Gong, and Mingyuan Zhou. Adversarially adaptive normalization for single domain generalization. In *Proceedings of the IEEE/CVF Conference on Computer Vision and Pattern Recognition (CVPR)*, pages 8208–8217, June 2021.

[16] Yaroslav Ganin, Evgeniya Ustinova, Hana Ajakan, Pascal Germain, Hugo Larochelle, François Laviolette, Mario Marchand, and Victor Lempitsky. Domain-adversarial training of neural networks. *The journal of machine learning research*, 17(1):2096–2030, 2016.

[17] Jin Gao, Jialing Zhang, Xihui Liu, Trevor Darrell, Evan Shelhamer, and Dequan Wang. Back to the source: Diffusion-driven adaptation to test-time corruption. In *Proceedings of the IEEE/CVF Conference on Computer Vision and Pattern Recognition*, pages 11786–11796, 2023.

[18] Theodoros Georgiou, Yu Liu, Wei Chen, and Michael Lew. A survey of traditional and deep learning-based feature descriptors for high dimensional data in computer vision. *International Journal of Multimedia Information Retrieval*, 9(3):135–170, 2020.

[19] Taesik Gong, Jongheon Jeong, Taewon Kim, Yewon Kim, Jinwoo Shin, and Sung-Ju Lee. Note: Robust continual test-time adaptation against temporal correlation. *Advances in Neural Information Processing Systems*, 35:27253–27266, 2022.

[20] Kaiming He, Xiangyu Zhang, Shaoqing Ren, and Jian Sun. Deep residual learning for image recognition. In *Proceedings of the IEEE conference on computer vision and pattern recognition*, pages 770–778, 2016.

[21] Dan Hendrycks, Steven Basart, Norman Mu, Saurav Kadavath, Frank Wang, Evan Dorundo, Rahul Desai, Tyler Zhu, Samyak Parajuli, Mike Guo, et al. The many faces of robustness: A critical analysis of out-of-distribution generalization. In *Proceedings of the IEEE/CVF international conference on computer vision*, pages 8340–8349, 2021.

[22] Dan Hendrycks and Thomas Dietterich. Benchmarking neural network robustness to common corruptions and perturbations. *arXiv preprint arXiv:1903.12261*, 2019.

[23] Dan Hendrycks, Kevin Zhao, Steven Basart, Jacob Steinhardt, and Dawn Song. Natural adversarial examples. In *Proceedings of the IEEE/CVF conference on computer vision and pattern recognition*, pages 15262–15271, 2021.

[24] Junyuan Hong, Lingjuan Lyu, Jiayu Zhou, and Michael Spranger. Mecta: Memory-economic continual test-time model adaptation. In *2023 International Conference on Learning Representations*, 2023.

[25] Jie Hu, Li Shen, and Gang Sun. Squeeze-and-excitation networks. In *Proceedings of the IEEE conference on computer vision and pattern recognition*, pages 7132–7141, 2018.

[26] Sergey Ioffe and Christian Szegedy. Batch normalization: Accelerating deep network training by reducing internal covariate shift. In *International conference on machine learning*, pages 448–456. pmlr, 2015.

[27] Yusuke Iwasawa and Yutaka Matsuo. Test-time classifier adjustment module for model-agnostic domain generalization. *Advances in Neural Information Processing Systems*, 34:2427–2440, 2021.

[28] Hemalatha Kanagaraj and V Muneeswaran. Image compression using haar discrete wavelet transform. In *2020 5th International Conference on Devices, Circuits and Systems (ICDCS)*, pages 271–274. IEEE, 2020.

[29] Hyun Kim, Albert No, and Hyuk-Jae Lee. Spiht algorithm with adaptive selection of compression ratio depending on dwt coefficients. *IEEE Transactions on Multimedia*, 20(12):3200–3211, 2018.

[30] Jan Kukačka, Vladimir Golkov, and Daniel Cremers. Regularization for deep learning: A taxonomy. *arXiv preprint arXiv:1710.10686*, 2017.

[31] Jogendra Nath Kundu, Naveen Venkat, R Venkatesh Babu, et al. Universal source-free domain adaptation. In *Proceedings of the IEEE/CVF Conference on Computer Vision and Pattern Recognition*, pages 4544–4553, 2020.

[32] Dong-Hyun Lee et al. Pseudo-label: The simple and efficient semi-supervised learning method for deep neural networks. In *Workshop on challenges in representation learning, ICML*, volume 3, page 896. Atlanta, 2013.

[33] Yoonho Lee, Annie S Chen, Fahim Tajwar, Ananya Kumar, Huaxiu Yao, Percy Liang, and Chelsea Finn. Surgical fine-tuning improves adaptation to distribution shifts. *arXiv preprint arXiv:2210.11466*, 2022.

[34] Haoliang Li, Sinno Jialin Pan, Shiqi Wang, and Alex C Kot. Domain generalization with adversarial feature learning. In *Proceedings of the IEEE conference on computer vision and pattern recognition*, pages 5400–5409, 2018.

[35] Lei Li, Ke Gao, Juan Cao, Ziyao Huang, Yepeng Weng, Xiaoyue Mi, Zhengze Yu, Xiaoya Li, and Boyang Xia. Progressive domain expansion network for single domain generalization. In *Proceedings of the IEEE/CVF Conference on Computer Vision and Pattern Recognition*, pages 224–233, 2021.

[36] Ming Li, Zheng Ma, Yu Guang Wang, and Xiaosheng Zhuang. Fast haar transforms for graph neural networks. *Neural Networks*, 128:188–198, 2020.

[37] Qiufu Li and Linlin Shen. Wavesnet: Wavelet integrated deep networks for image segmentation. In *Pattern Recognition and Computer Vision: 5th Chinese Conference, PRCV 2022, Shenzhen, China, November 4–7, 2022, 2022, Proceedings, Part IV*, pages 325–337. Springer, 2022.

[38] Qiufu Li, Linlin Shen, Sheng Guo, and Zhihui Lai. Wavecnet: Wavelet integrated cnns to suppress aliasing effect for noise-robust image classification. *IEEE Transactions on Image Processing*, 30:7074–7089, 2021.

[39] Jian Liang, Ran He, and Tieniu Tan. A comprehensive survey on test-time adaptation under distribution shifts. *arXiv preprint arXiv:2303.15361*, 2023.

[40] Yuejiang Liu, Parth Kothari, Bastien Van Delft, Baptiste Bellot-Gurlet, Taylor Mordan, and Alexandre Alahi. Ttt++: When does self-supervised test-time training fail or thrive? *Advances in Neural Information Processing Systems*, 34:21808–21820, 2021.

[41] Sachin Mehta and Mohammad Rastegari. Mobilevit: light-weight, general-purpose, and mobile-friendly vision transformer. *arXiv preprint arXiv:2110.02178*, 2021.

[42] Zachary Nado, Shreyas Padhy, D Sculley, Alexander D'Amour, Balaji Lakshminarayanan, and Jasper Snoek. Evaluating prediction-time batch normalization for robustness under covariate shift. *arXiv preprint arXiv:2006.10963*, 2020.

[43] Vinod Nair and Geoffrey E Hinton. Rectified linear units improve restricted boltzmann machines. In *Proceedings of the 27th international conference on machine learning (ICML-10)*, pages 807–814, 2010.

[44] Xuan Truong Nguyen, Hyuk-Jae Lee, and Hyun Kim. A low-cost hardware design of a 1-d spiht algorithm for video display systems. *IEEE Transactions on Consumer Electronics*, 64(1):44–52, 2018.

[45] Shuaicheng Niu, Jiaxiang Wu, Yifan Zhang, Yaofo Chen, Shijian Zheng, Peilin Zhao, and Mingkui Tan. Efficient test-time model adaptation without forgetting. In *International conference on machine learning*, pages 16888–16905. PMLR, 2022.

[46] Shuaicheng Niu, Jiaxiang Wu, Yifan Zhang, Zhiquan Wen, Yaofo Chen, Peilin Zhao, and Mingkui Tan. Towards stable test-time adaptation in dynamic wild world. *arXiv preprint arXiv:2302.12400*, 2023.

[47] Mostafa Parchami, Saman Bashbaghi, and Eric Granger. Video-based face recognition using ensemble of haar-like deep convolutional neural networks. In *2017 International Joint Conference on Neural Networks (IJCNN)*, pages 4625–4632. IEEE, 2017.

[48] Viraj Prabhu, Shivam Khare, Deeksha Kartik, and Judy Hoffman. Augco: augmentation consistency-guided self-training for source-free domain adaptive semantic segmentation. *arXiv preprint arXiv:2107.10140*, 2021.

[49] Fengchun Qiao, Long Zhao, and Xi Peng. Learning to learn single domain generalization. In *Proceedings of the IEEE/CVF Conference on Computer Vision and Pattern Recognition (CVPR)*, June 2020.

[50] Sriprabha Ramanarayanan, Balamurali Murugesan, Keerthi Ram, and Mohanasankar Sivaprakasam. Dc-wcnn: A deep cascade of wavelet based convolutional neural networks for mr image reconstruction. In *2020 IEEE 17th International Symposium on Biomedical Imaging (ISBI)*, pages 1069–1073. IEEE, 2020.

[51] K Ramamohan Rao and Ping Yip. *Discrete cosine transform: algorithms, advantages, applications*. Academic press, 2014.

[52] Steffen Schneider, Evgenia Rusak, Luisa Eck, Oliver Bringmann, Wieland Brendel, and Matthias Bethge. Improving robustness against common corruptions by covariate shift adaptation. *Advances in neural information processing systems*, 33:11539–11551, 2020.

[53] Skyler Seto, Barry-John Theobald, Federico Danieli, Navdeep Jaitly, and Dan Busbridge. Realm: Robust entropy adaptive loss minimization for improved single-sample test-time adaptation. In *Proceedings of the IEEE/CVF Winter Conference on Applications of Computer Vision*, pages 2062–2071, 2024.

[54] Jin Shin and Hyun Kim. Rl-spiht: Reinforcement learning-based adaptive selection of compression ratios for 1-d spiht algorithm. *IEEE Access*, 9:82485–82496, 2021.

[55] Junha Song, Jungsoo Lee, In So Kweon, and Sungha Choi. Ecotta: Memory-efficient continual test-time adaptation via self-distilled regularization. In *Proceedings of the IEEE/CVF Conference on Computer Vision and Pattern Recognition*, pages 11920–11929, 2023.

[56] Dmitry Ulyanov, Andrea Vedaldi, and Victor Lempitsky. Instance normalization: The missing ingredient for fast stylization. *arXiv preprint arXiv:1607.08022*, 2016.

[57] Vladimir Vapnik. Principles of risk minimization for learning theory. *Advances in neural information processing systems*, 4, 1991.

[58] Riccardo Volpi, Hongseok Namkoong, Ozan Sener, John C Duchi, Vittorio Murino, and Silvio Savarese. Generalizing to unseen domains via adversarial data augmentation. *Advances in neural information processing systems*, 31, 2018.

[59] Dequan Wang, Evan Shelhamer, Shaoteng Liu, Bruno Olshausen, and Trevor Darrell. Tent: Fully test-time adaptation by entropy minimization. *arXiv preprint arXiv:2006.10726*, 2020.

[60] Haohan Wang, Songwei Ge, Zachary Lipton, and Eric P Xing. Learning robust global representations by penalizing local predictive power. *Advances in Neural Information Processing Systems*, 32, 2019.

[61] Haohan Wang, Xindi Wu, Zeyi Huang, and Eric P Xing. High-frequency component helps explain the generalization of convolutional neural networks. In *Proceedings of the IEEE/CVF conference on computer vision and pattern recognition*, pages 8684–8694, 2020.

[62] Qin Wang, Olga Fink, Luc Van Gool, and Dengxin Dai. Continual test-time domain adaptation. In *Proceedings of the IEEE/CVF Conference on Computer Vision and Pattern Recognition*, pages 7201–7211, 2022.

[63] Garrett Wilson and Diane J Cook. A survey of unsupervised deep domain adaptation. *ACM Transactions on Intelligent Systems and Technology (TIST)*, 11(5):1–46, 2020.

[64] Xiang Xu, Xiong Zhou, Ragav Venkatesan, Gurumurthy Swaminathan, and Orchid Majumder. d-sne: Domain adaptation using stochastic neighborhood embedding. In *Proceedings of the IEEE/CVF Conference on Computer Vision and Pattern Recognition*, pages 2497–2506, 2019.

[65] Marvin Zhang, Sergey Levine, and Chelsea Finn. Memo: Test time robustness via adaptation and augmentation. *Advances in Neural Information Processing Systems*, 35:38629–38642, 2022.

[66] Kaiyang Zhou, Yongxin Yang, Timothy Hospedales, and Tao Xiang. Deep domain-adversarial image generation for domain generalisation. In *Proceedings of the AAAI Conference on Artificial Intelligence*, volume 34, pages 13025–13032, 2020.

## A  Appendix: A Detailed Implementation Setting of the Proposed Models

The image classification process includes applying a reconstructed stem layer followed by 10 epochs of warm-up training using cross-entropy loss [57]. In both TTA and warm-up settings, the batch size and learning rate are set to 128 and 0.05, respectively. For CIFAR-10 and CIFAR-100, the model utilizes weights pre-trained on ImageNet and is trained for 150 epochs, using the same batch size and learning rate configuration as in the ImageNet training setup. The optimizer used is standard SGD [1], consistent across all datasets. This indicates the non-necessity for additional search for hyperparameters in real fields. The only different parameter is the reduction size of the SE block, which was determined experimentally through ablation studies in Appendix D. CIFAR-10-C and CIFAR-100-C have a relatively small number of classes and are assigned a reduction of eight. ImageNet-C is assigned a reduction of 32.

To evaluate the performance of TTA in semantic segmentation, Cityscapes-C was generated by synthesizing four weather corruptions with a severity of 5 out of 15 corruptions in the Cityscapes dataset [13]. Additionally, for an unbiased comparison with the settings of EcoTTA [55], we use the DeepLabV3Plus model [7] with the backbone of ResNet-50 and pre-train this model with the instance selective whitening strategy proposed in RobustNet [10]. To warm up the stem layer, 32,000 iterations are trained with Cityscapes with a batch size of 8. For TTA, Cityscapes-C is learned with a batch size of 2, and the performance metrics mIoU and memory usage are compared with those of existing methods.

The source code is available for reproducibility at the GitHub link provided in the abstract. The seed for random number generator is commonly used as 42. However, please note that this does not significantly affect the accuracy of the model. We conducted experiments on 15 main corruptions using 10 additional seeds (*i.e.,* 3, 10, 21, 22, 43, 99, 318, 500, 565, 777), along with the previously used seed (42). Each experiment was run for 10 iterations in a single batch per seed, resulting in an average accuracy of 64.85% with a standard deviation of 0.02. All experiments were conducted on a system equipped with an Intel® Xeon® Gold 5218R CPU and an NVIDIA Tesla A100 80G GPU. However, the minimum GPU memory required to conduct the single experiment was measured to be approximately 12GB.

## B  Appendix: Operation Process of 2-D Discrete Wavelet Transform

The 2D DWT involves sequentially applying a 1D DWT to the rows and columns of the input data. For example, in the case of the Haar wavelet [36], the low and high filter banks are defined as follows:

$$\psi_k^+ = \left\{ \frac{1}{\sqrt{2}}, \frac{1}{\sqrt{2}} \right\}, \quad \psi_k^- = \left\{ \frac{1}{\sqrt{2}}, -\frac{1}{\sqrt{2}} \right\}. \tag{7}$$

In the first stage, it is formulated in the horizontal direction decomposition as follows:

$$L_{i,j} = \sum_{n=0}^{H-1} \sum_{m=0}^{W-1} (X_{2m,n} * \psi_0^+ + X_{2m+1,n} * \psi_1^+), H_{i,j} = \sum_{n=0}^{H-1} \sum_{m=0}^{W-1} (X_{2m,n} * \psi_0^- + X_{2m+1,n} * \psi_1^-).$$
$$\tag{8}$$

This describes the splitting of a single original feature map into two components: the LFC and the HFC. Here, $X_{m,n}$ denotes the pixels of the input image. $W$ and $H$ denote the width and height of $X$, respectively. $*$ denotes the CONV. From a CONV perspective, the two filter banks denote the kernel weights, and $m$ and $n$ denote the stride in the horizontal and vertical directions, respectively.

In the second stage, a vertical decomposition is performed as the two filters are transposed to operate in Eq. 8. The exception is that each $X$ is replaced by $L$ and $H$, respectively, and because it is decomposed vertically for $L$ and $H$, the indices $m$ and $n$ are interchanged in Eq. 8. Because of the decomposition, $L$ is divided into $LH$ and $HL$, and $H$ is decomposed into $HL$ and $HH$. The LFC ($LL$) generally contains visually intuitive information such as the overall image brightness. It exists in the range [0, $+2P_{max}$]. Herein, $P_{max}$ is the maximum value of image pixels (8 bits = 255). This area's information represents the basic image structure. It is significantly correlated with the HFCs. Meanwhile, HFCs (*i.e.*, $LH$, $HL$, $HH$) include finer details of the image and a small amount of edge information including information regarding diagonal variations and the image texture or pattern. The range of values is within [-$P_{max}/2$, $+P_{max}/2$]. These obtained frequency domains have been variously analyzed and applied for image processing purposes [38, 50, 37].

## C   Appendix: Rigorous Assessments of Model Robustness

To demonstrate that the proposed method can adapt not only to the introduced corruptions but also to challenging domain shifts, we conduct experiments on three well-known rigorous datasets with challenging themes separated from ImageNet. This experiment is reported as prediction error (%), the same as all experiments. As shown in Tab. 3, it is almost impossible to have classification ability on the test datasets of Imagenet-Sketch [21], ImageNet-R [60] and ImageNet-A [23] even though they have already been pre-trained with the same domain. However, this can be alleviated by applying the proposed DEL structure to ResNet-50, and when TTA is performed in real time with the proposed method, the model accuracy can be improved to 2.84%, 40.91%, and 5.49% for each ImageNet series.

Table 3: Evaluation of the proposed method using ResNet-50 on challenging ImageNet series.

| Dataset | Source | Ours w/o GCAL | Ours |
|---|---|---|---|
| ImageNet-Sketch[21] | 75.84 | 73.93 | **73.00 (-2.84)** |
| ImageNet-R[60] | 99.43 | 62.77 | **58.52 (-40.91)** |
| ImateNet-A[23] | 99.58 | 94.35 | **94.09 (-5.49)** |

## D   Appendix: Ablation Studies on Architectural Exploration

In our experiments, we assign a DWT level of two for DEL as mentioned in Sec. 3.3. To experimentally demonstrate the advantages of ODD, in Tab. 4, we describe the results according to the DWT level on the CIFAR-10-C, CIFAR-100-C, and ImageNet-C datasets. To summarize, providing a more varied multi-view through ODD rather than only decomposing one time with DWT yields higher generalization performance and prediction accuracy in TTA. The average improvement is 1.5%.

Table 4: Comparison of TTA results in terms of average prediction errors for source, DWT level 1, and level 2 decompositions using ResNet-50 on CIFAR-10-C, CIFAR-100-C, and ImageNet-C.

Table 5: Comparison TTA results in terms of average prediction errors using ResNet-50 on CIFAR-10-C, CIFAR-100-C, and ImageNet-C, focusing on the influence of the SE block's reduction scale.

| Method | CIFAR-10-C | CIFAR-100-C | ImageNet-C |
|---|---|---|---|
| Source | 36.6 | 53.6 | 82.0 |
| DWT Level 1 | 33.3 | 52.4 | 72.9 |
| + TTA | 20.5 | 39.3 | 65.2 |
| DWT Level 2 | 30.8 | 50.8 | 72.6 |
| + TTA | **18.0** | **37.5** | **64.9** |

| REDUCTION | CIFAR-10-C | CIFAR-100-C | IMAGENET-C |
|---|---|---|---|
| 4 | 18.0 | **37.5** | 67.4 |
| 8 | **17.9** | 39.2 | 67.2 |
| 16 | 19.2 | 38.2 | 66.5 |
| 32 | 18.0 | 39.4 | **64.9** |

Tab. 5 presents an ablation study on the reduction size, a hyperparameter of the SE block that constitutes GCAL. In the three datasets used for the experiments (*i.e.*, CIFAR-10-C, CIFAR-100-C, and ImageNet-C datasets), 8, 4, and 32 display the best results, respectively. However, no significant difference exists except when the reduction size is 16 in CIFAR-10-C. It should be noted that regardless of the reduction size, we exceed the SOTA results. Meanwhile, for large datasets with many classes such as ImageNet-C, minimizing the reduction appears to yield linearly better advantages. Therefore, we empirically assign a reduction size of 32 for ImageNet-C.

## E   Appendix: Further Case Studies

In Sec. 4, we compare our results primarily with studies that show significant improvements in accuracy by performing complicated processes. In Tab. 1, we demonstrate that our accuracy is reasonable by presenting the prediction errors on CIFAR-10-C and ImageNet-C using ResNet-26 and ResNet-50 models, respectively. For a comprehensive performance comparison, including memory usage, Tab. 6 compares our proposal on CIFAR-10-C and CIFAR-100-C using ResNet-50 against existing TTA methodologies. Notably, it shows superiority even over methods like CoTTA, which

Table 6: Comparisons of prediction error (%) with prior TTA methods on ResNet-50.

| Method | Gauss. | Shot | Impul. | Defoc. | Glass | Motion | Zoom | Snow | Frost | Fog | Brit. | Contr. | Elastic | Pixel | JPEG | Avg. | Mem. (MB) |
|---|---|---|---|---|---|---|---|---|---|---|---|---|---|---|---|---|---|
| | | | | | | **CIFAR-10-C** | | | | | | | | | | | |
| Source | 65.6 | 60.7 | 74.4 | 28.9 | 79.9 | 46.0 | 25.7 | 35.0 | 49.4 | 54.7 | 13.0 | 83.2 | 41.2 | 46.7 | 27.7 | 48.8 | - |
| BN STAT[42] | 18.0 | 17.2 | 29.3 | 10.7 | 27.2 | 15.5 | 8.9 | 16.7 | 14.6 | 21.0 | 9.3 | 12.7 | 20.9 | 12.4 | 14.8 | 16.6 | 91 |
| TENT[59] | 16.6 | 15.7 | 25.7 | 10.0 | 24.8 | 13.8 | 8.3 | 14.9 | 13.8 | 17.6 | 8.7 | 10.0 | 19.1 | 11.5 | 13.8 | 15.0 | 925 |
| TTT++[40] | 18.2 | 16.9 | 28.7 | 10.5 | 26.5 | 14.5 | 8.9 | 16.5 | 14.5 | 20.9 | 9.0 | 9.0 | 20.4 | 12.3 | 14.7 | 16.1 | 1877 |
| SWRNSP[11] | 17.3 | 16.1 | 26.1 | 10.6 | 25.6 | 14.1 | 8.7 | 15.6 | 13.6 | 18.6 | 8.8 | 10.0 | 19.3 | 12.0 | 14.2 | 15.4 | 1971 |
| EATA[45] | 17.2 | 14.9 | 23.6 | 10.2 | 23.3 | 13.2 | 8.5 | 14.0 | 12.5 | 16.6 | 8.6 | 9.4 | 17.2 | 11.0 | 12.7 | 14.2 | 925 |
| CoTTA[62] | 16.2 | 15.0 | 21.2 | 10.4 | 22.8 | 13.9 | 8.4 | 15.1 | 12.9 | 19.8 | 8.6 | 11.3 | 17.5 | **10.5** | **12.2** | 14.4 | 2066 |
| EcoTTA (Mem+)[55] | **16.5** | 14.5 | 24.3 | 9.7 | 23.7 | 13.3 | 8.8 | 14.7 | 12.9 | 17.0 | 9.1 | 9.4 | 17.6 | 11.4 | 13.1 | 14.4 | 296 |
| EcoTTA (Acc+)[55] | 16.6 | **14.4** | 23.6 | 9.8 | 23.4 | 12.7 | 8.6 | 14.5 | 12.6 | 16.6 | 8.7 | 9.0 | **17.0** | 11.3 | 12.6 | **14.1** | 498 |
| Ours | 22.4 | 18.8 | 28.3 | **7.8** | 30.4 | **9.9** | **6.5** | **8.6** | **9.3** | **9.2** | **4.2** | **5.6** | 17.3 | 11.6 | 22.1 | **14.1** | **6.4** |
| | | | | | | **CIFAR-100-C** | | | | | | | | | | | |
| Source | 84.7 | 83.5 | 93.3 | 59.6 | 92.5 | 71.9 | 54.8 | 66.6 | 77.6 | 81.8 | 44.3 | 91.2 | 72.2 | 76.6 | 56.5 | 73.8 | - |
| BN STAT[42] | 48.1 | 46.7 | 60.6 | 35.1 | 58.0 | 41.8 | 33.2 | 47.3 | 43.5 | 54.9 | 33.5 | 35.3 | 49.8 | 38.4 | 40.8 | 44.5 | 91 |
| TENT[59] | 44.1 | 42.7 | 53.9 | 32.6 | 52.0 | 37.5 | 30.5 | 43.4 | 40.2 | 45.7 | 30.4 | 31.4 | 45.1 | 35.0 | 37.6 | 40.1 | 926 |
| TTT++[40] | 48.1 | 46.5 | 60.8 | 35.1 | 57.8 | 41.6 | 32.9 | 46.8 | 43.3 | 55.0 | 33.3 | 34.0 | 50.0 | 38.1 | 40.6 | 44.3 | 1876 |
| SWRNSP[11] | 48.3 | 46.5 | 60.5 | 35.1 | 57.9 | 41.7 | 32.9 | 47.1 | 43.5 | 54.7 | 33.5 | 35.1 | 49.9 | 38.3 | 40.7 | 44.4 | 1970 |
| EATA[45] | 44.8 | 41.9 | 52.6 | 33.0 | 51.1 | 37.8 | 30.3 | 43.0 | 40.1 | 45.1 | 30.1 | 31.8 | 45.2 | 35.2 | 37.4 | 40.0 | 926 |
| CoTTA[62] | **43.6** | 42.8 | 50.4 | 34.2 | 51.6 | 39.2 | 31.4 | 43.4 | 39.6 | 47.4 | 31.3 | 32.2 | 43.4 | 35.8 | **36.7** | 40.2 | 2064 |
| EcoTTA (Mem+)[55] | 44.8 | **40.3** | 49.2 | 32.3 | 50.1 | 36.3 | 29.5 | 41.0 | 39.9 | 44.6 | 31.5 | 33.7 | 45.3 | 36.3 | 37.7 | 39.5 | 296 |
| EcoTTA (Acc+)[55] | 44.9 | 40.4 | **48.9** | 32.7 | **49.7** | 36.9 | 29.3 | 40.8 | 39.0 | 44.4 | 31.1 | 33.6 | 44.0 | 35.7 | 37.8 | 39.3 | 498 |
| Ours | 52.4 | 49.9 | 55.7 | **25.8** | 56.8 | **27.4** | **23.6** | **31.2** | **30.7** | **31.9** | **18.6** | **23.5** | 41.2 | **31.8** | 50.6 | **36.7** | **6.4** |

maintain dual models during training. Additionally, our method improves the average prediction error by 2.6% while using 98.71% less memory compared to EcoTTA, which is designed to use more memory to enhance accuracy on CIFAR-100-C.

In all experiments of this paper, results are presented using ResNet. To address concerns regarding the applicability to other architectures, we reconstruct the stem layer of the MobileViT[41], which combines a vision transformer and a CONV structure, using our proposed method to perform TTA. We use MobileViT-XXS, the most suitable model for mobile environments, which sets the kernel size to $3\times3$, unlike the vanilla ResNet series. As shown in Tab. 7, we conduct an ablation study for three cases (*i.e.*, Ours w/o GCAL, Ours w/o DEL, and Ours) and observe similar trends to those in Tab 1. By applying the proposed stem layer, the average accuracy is improved by 15.7%.

Additionally, we discuss two studies similar to our approach and compare performance with source-free DA (SFDA) studies similar to the TTA scenario. SFDA generates learning criteria such as pseudo-labels or prototypes to update the target model, requiring dual maintenance of a well-trained model, resulting in higher memory consumption compared to TTA. To address this, T3A [27] and LAME [3] propose classifier adjustment-based methods similar to our approach. However, since they only update the last layer, the learning cost for criterion generation and forwarding is continuously required. Tab. 8 and 9 compare our experimental results with similar approaches, representative TTAs, and SFDA studies on both CIFAR-10-C and CIFAR-100-C using ResNet-50 and ResNet-18, demonstrating better performance with extremely low memory consumption. It should be noted that the SFDA scenario consumes approximately twice as much memory as TENT [59].

Table 7: Ablation Study of Prediction Errors (%) on the CIFAR-10-C using MobileViT-XXS.

| Method | Gauss. | Shot | Impul. | Defoc. | Glass | Motion | Zoom | Snow | Frost | Fog | Brit. | Contr. | Elastic | Pixel | JPEG | Avg. |
|---|---|---|---|---|---|---|---|---|---|---|---|---|---|---|---|---|
| Source | 78.9 | 75.4 | 78.1 | 11.3 | 49.3 | 15.9 | 8.5 | 13.7 | 23.4 | 17.6 | 6.1 | 15.6 | 20.6 | 50.6 | 25.7 | 32.7 |
| Ours w/o GCAL | 78.0 | 73.6 | 79.9 | 10.6 | 48.9 | 16.2 | 8.1 | 13.4 | 22.5 | 17.6 | 6.4 | 16.0 | 20.2 | 24.9 | 24.9 | 30.8 |
| Ours w/o DEL | 34.2 | 29.0 | 32.6 | 8.8 | 33.2 | 10.6 | 8.5 | 12.2 | 13.9 | 11.8 | 6.0 | 8.7 | 18.3 | 14.9 | 26.7 | 18.0 |
| Ours | **32.3** | **27.9** | **41.2** | **8.1** | **30.4** | **10.5** | **7.3** | **10.0** | **11.6** | **11.3** | **5.6** | **8.4** | **16.4** | **10.7** | **23.7** | **17.0** |

Table 8: Comparison of Prediction Errors (%) between the proposed method and high-cost methodologies on ResNet-50.

| Method | CIFAR-10-C | CIFAR-100-C |
|---|---|---|
| Source | 29.15 | 60.34 |
| TENT[59] | 14.27 | 40.72 |
| T3A[27] | 15.44 | 42.72 |
| CoTTA[62] | 14.4 | 40.2 |
| Ours | **14.1** | **36.7** |

Table 9: Comparison of Prediction Errors (%) between the proposed method and high-cost methodologies on ResNet-18.

| Method | CIFAR-10-C | CIFAR-100-C |
|---|---|---|
| Source | 42.3 | 66.6 |
| Pseudo-Label[32] | 21.6 | 43.1 |
| TENT[59] | 18.8 | 40.3 |
| LAME[3] | 44.1 | 68.8 |
| CoTTA[62] | 17.8 | 44.3 |
| NOTE[19] | 17.6 | 41.0 |
| Ours | **15.8** | **39.5** |

# F    Appendix: Limitations and Discussions

Our proposed lightweight TTA approach reconstructs the stem layer to extract the channel-wise uncertainty and updates the parameters before the BN layer by minimizing the uncertainty. It quickly adapts the initial representation to be robust to $D_t$ by consuming less memory by not learning subsequent layers. However, it can be observed that the stem layer has less impact on improving the robustness of the model as the number of classes increases, as shown in the experimental result of this paper (*e.g.*, Tab. 1). This is considered a curse of dimensionality. Furthermore, Fig. 8 shows the mean prediction errors of four corruption themes (*i.e.*, noise, blur, weather, and digital) as the number of training iterations increases. Here we do not apply the interval training strategy proposed in Sec. 3.4. As shown in ImageNet-C, the iteration that yields the best performance for each theme is different, and continuously minimizing uncertainty will cause the model to diverge. To prevent this, we perform interval training. These observations are not observed in datasets with a relatively small number of classes like CIFAR-100-C, which converges stably even without interval training. Moreover, as described in Tab. 2, we demonstrate its practicality by showing sufficient performance improvement in autonomous driving scenarios such as semantic segmentation.

To address these challenges, several unexplored potential solutions could be considered. The issue of handling a large number of classes, as identified in this study, may be mitigated by applying the channel attention uncertainty minimization strategy not only in the stem layer but also selectively in subsequent layers. Additionally, the overfitting-related regularization problem could be tackled by actively filtering out data to enhance generalization performance, as seen in methods like EATA[45] and REALM[53], or by adapting to input data rather than unconditionally minimizing uncertainty. However, such an approach should be designed to enable the network to autonomously adjust to independent data, aligning with our research philosophy.

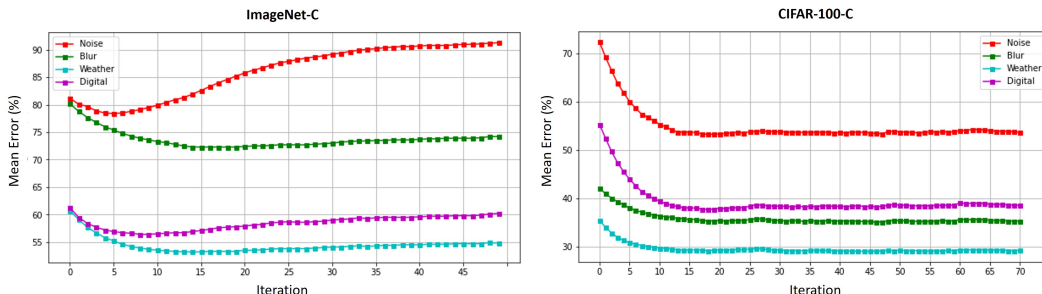

Figure 8: Mean error rates (%) by corruption type on ImageNet-C and CIFAR-100-C with increasing TTA iterations in ResNet-50.

